# Detection of first and second order motion

**Alexander Grunewald**
Division of Biology
California Institute of Technology
Mail Code 216-76
Pasadena, CA 91125
alex@vis.caltech.edu

**Heiko Neumann**
Abteilung Neuroinformatik
Universität Ulm
89069 Ulm
Germany
hneumann@neuro.informatik.uni-ulm.de

## Abstract

A model of motion detection is presented. The model contains three stages. The first stage is unoriented and is selective for contrast polarities. The next two stages work in parallel. A phase *insensitive* stage pools across different contrast polarities through a spatiotemporal filter and thus can detect first and second order motion. A phase *sensitive* stage keeps contrast polarities separate, each of which is filtered through a spatiotemporal filter, and thus only first order motion can be detected. Differential phase sensitivity can therefore account for the detection of first and second order motion. Phase insensitive detectors correspond to cortical complex cells, and phase sensitive detectors to simple cells.

## 1   INTRODUCTION

In our environment objects are constantly in motion, and the visual system faces the task of identifying the motion of objects. This task can be subdivided into two components: motion detection and motion integration. In this study we will look at motion detection. Recent psychophysics has made a useful distinction between first and second order motion. In first order motion an absolute image feature is moving. For example, a bright bar moving on a dark background is an absolute feature because luminance is moving. In second order motion a relative image feature is moving, for example a contrast reversing bar. No longer is it possible to identify the moving object through its luminance, but only that it has different luminance with respect to the background. Humans are very sensitive to first order motion, but can they detect second order motion? Chubb & Sperling (1988) showed that subjects are in fact able to detect second order motion. These findings have since been confirmed in many psychophysical experiments, and it has become clear that the parameters that yield detection of first and second order motion are different, suggesting that separate motion detection systems exist.

## 1.1   Detection of first and second order motion

First order motion, which is what we encounter in our daily lives, can be easily detected by finding the peak in the Fourier energy distribution. The motion energy detector developed by Adelson & Bergen (1985) does this explicitly, and it turns out that it is also equivalent to a Reichardt detector (van Santen & Sperling, 1985). However, these detectors cannot adequately detect second order motion, because second order motion stimuli often contain the maximum Fourier energy in the opposite direction (possibly at a different velocity) as the actual motion. In other words, purely linear filters, should have opposite directional tuning for first and second order motion. This is further illustrated in Figure 1.

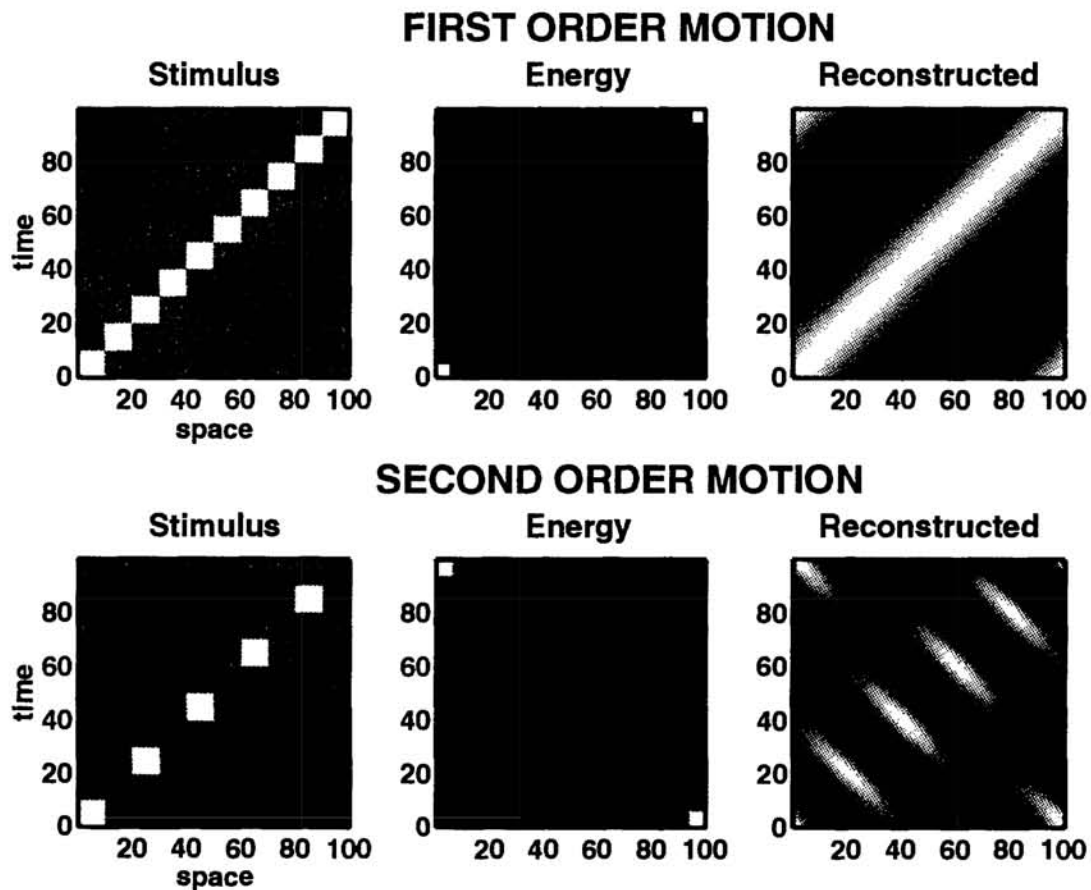

Figure 1: Schematic of first and second order motion, their peak Fourier energy, and the reconstruction. The peak Fourier energy is along the direction of motion for first order motion, and in the opposite direction for second order motion. For this reason a linear filter cannot detect second order motion.

One way to account for second order motion detection is to transform the second order motion signal into a first order signal. If second order motion is defined by contrast reversals, then detecting contrast edges and then rectifying the resulting signal of contrast will yield a first order motion signal. Thus this approach includes three steps: orientation detection, rectification and finally motion detection (Wilson *et al.*, 1992).

## 1.2 Visual physiology

Cells in the retina and the lateral geniculate nucleus (LGN) have concentric (and hence unoriented) receptive fields which are organized in an opponent manner. While the center of such an ON cell is excited by a light increment, the surround is excited by a light decrement, and vice versa for OFF cells. It is only at the cortex that direction and orientation selectivity arise. Cortical simple cells are sensitive to the phase of the stimulus, while complex cells are not (Hubel & Wiesel, 1962).

Most motion models take at least partial inspiration from known physiology and anatomy, by relating the kernels of the motion detectors to the physiology of cortical cells. The motion energy model in particular detects orientation and first order motion at the same time. Curiously, all motion models essentially ignore the concentric opponency of receptive fields in the LGN. This is usually justified by pointing to the linearity of simple cells with respect to stimulus parameters. However, it has been shown that simple cells in fact exhibit strong nonlinearities (Hammond & MacKay, 1983). Moreover, motion detection does require at least one stage of nonlinearity (Poggio & Reichardt, 1973). The present study develops a model of first and second order motion detection which explicitly includes an unoriented processing stage, and phase sensitive and phase insensitive motion detectors are built from these unoriented signals. The former set of detectors only responds to first order motion, while the second set of detectors responds to both types of motion. We further show the analogies that can be drawn between these detector types and simple and complex cells in cat visual cortex.

## 2  MODEL DESCRIPTION

The model is two-dimensional, one dimension is space, which means that space has been collapsed onto a line, and the other dimension is time. The input image to the model is a space-time matrix of luminances, as shown in figure 1. At each processing stage essentially the same operations are performed. First the input signal is convolved with the appropriate kernel. At each stage there are multiple kernels, to generate the different signal types at that stage. For example, there are ON and OFF signals at the unoriented stage. Next the convolved responses are subtracted from each other. At the unoriented stage this means ON-OFF and OFF-ON. In the final step these results are half-wave rectified to only yield positive signals.

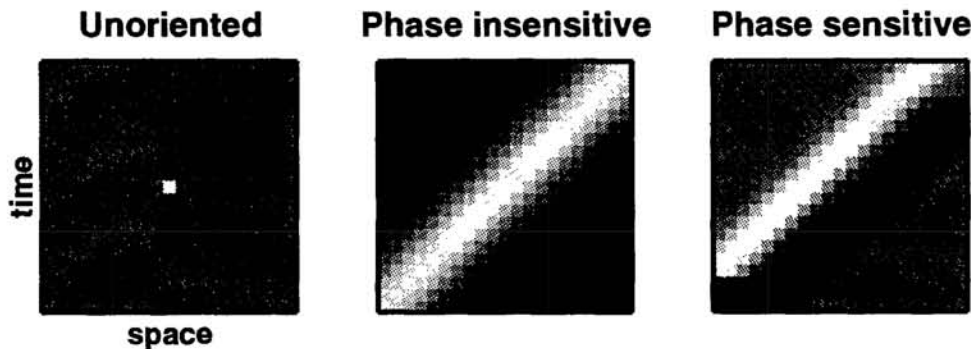

**Unoriented**     **Phase insensitive**     **Phase sensitive**

Figure 2: The kernels in the model. For the unoriented (left plot) and phase sensitive (right plot) kernel plots black indicates OFF regions, white ON regions, and grey zero input. For the phase insensitive plot (middle) grey denotes ON and OFF input, and black denotes zero input.

At the unoriented stage the input pattern is convolved with a difference of Gaussians kernel. This kernel has only a spatial dimension, no temporal dimension (see figure 2). As described earlier, competition is between ON and OFF signals, followed by half-wave rectification. This ensures that at each location only one set of unoriented signals is present. A simulation of the signals at the unoriented stage is shown in figure 3. For first order motion, ON signals are at locations corresponding to the inside of the moving bar. With each shift of the bar the signals also move. Similarly, the OFF signals correspond to the outside of the bar, and also move with the bar. For second order motion the contrast polarity reverses. Thus ON signals correspond to the inside when the bar is bright, and to the outside when the bar is dark, and vice versa for OFF signals. Thus any ON or OFF signals to the leading edge of the bar will remain active after the bar moves.

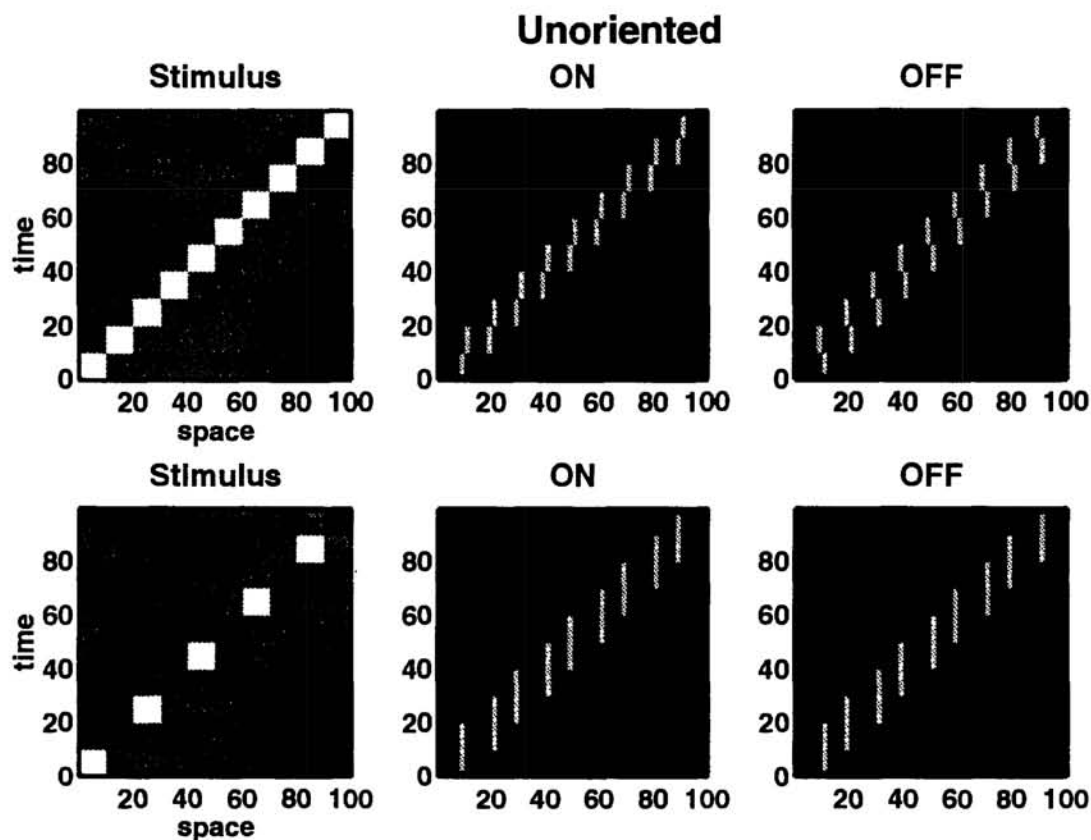

Figure 3: Unoriented signals to first and second order motion. ON signals are at the bright side of any contrast transition, while OFF signals are at the dark side. In first order motion ON and OFF move synchronously to the moving stimulus. In second order motion ON and OFF signals persist, since the leading edge becomes the trailing edge, and at the same time the contrast reverses, which means that at a particular spatial location the contrast remains constant.

At the phase insensitive stage the unoriented ON and OFF signals are added, and then the result is convolved with an energy detection filter. The pooling of ON and OFF signals means that the contrast transitions in the image are essentially full-wave rectified. This causes phase insensitivity. These pooled signals are then convolved with a space-time oriented filter (see figure 2). Competition between opposite directions of motion ensures that only one direction is active. A consequence of the pooling of unoriented ON and OFF signals at this stage is that the resulting signals are invariant to first or second order motion. Thus phase insensitivity

makes this stage able to detect both first and second order motion. These signals are shown in figure 4. In a two-dimensional extension of this model these detectors would also be orientation selective. The simplest way to obtain this would be via elongation along the preferred orientation.

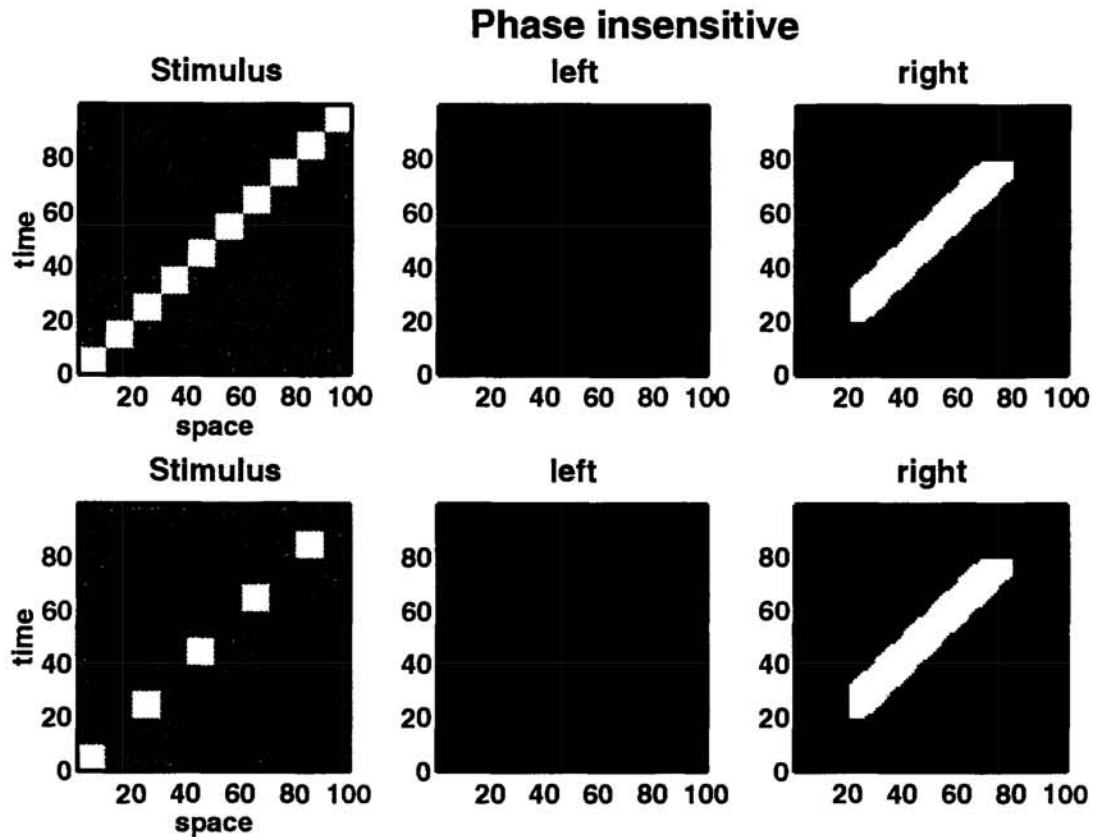

Figure 4: Phase insensitive signals to first and second order motion. For both stimuli there are no leftwards signals, and robust rightwards signals.

At the phase sensitive stage unoriented ON and OFF signals are separately convolved with space-time oriented kernels which are offset with respect to each other (see figure 2). The separate treatment of ON and OFF signals yields phase sensitivity. At each location there are four kernels: two for the two directions of motion, and two for the two phases. Competition occurs between signals of opposite direction tuning, and opposite phase preference. To avoid activation in the opposite direction of motion slightly removed from the location of the edge spatially broadly tuned inhibition is necessary. This is provided by the phase insensitive signals, thus avoiding feedback loops among phase sensitive detectors. First order signals from the unoriented stage match the spatiotemporal filters in the preferred direction, and thus phase sensitive signals arise. However, due to their phase reversal, second order motion input, provides poor motion signals, which are quenched through phase insensitive inhibition. These signals are shown in figure 5.

These simulations show that first and second order motion are detected differently. First order motion is detected by phase sensitive and phase insensitive motion detectors, while second order motion is only detected by the latter. From this we conclude that first order motion is a more potent stimulus, and that the detection of second order is more restricted, since it depends on a single type of detector. In particular, the size of the stimulus and its velocity have to be matched to the energy

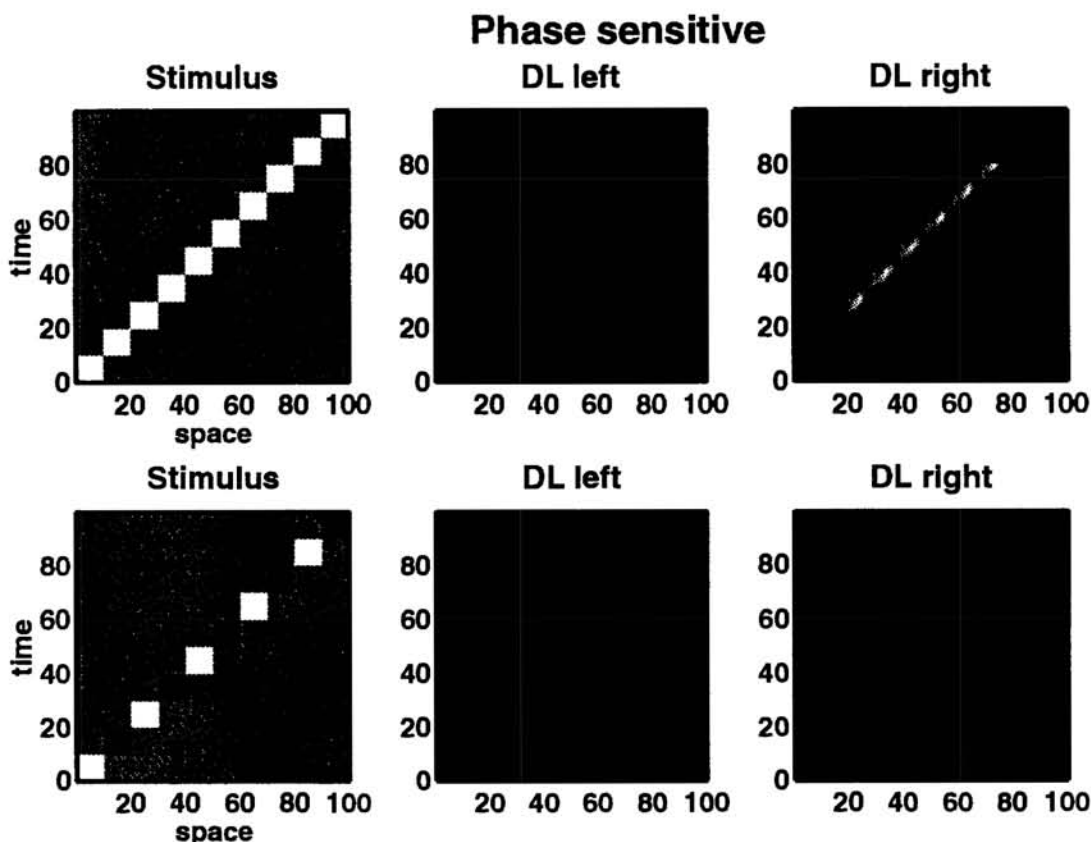

Figure 5: Phase sensitive signals to first and second order motion. Only the dark-light signals are shown. First order motion causes a consistent rightward motion signal, while second order motion does not.

filters for motion signals to arise.

## 3   RELATION TO PHYSIOLOGY

The relationship between the model and physiology is straightforward. Unoriented signals correspond to LGN responses, phase insensitive signals to complex cell responses, and phase sensitive signals to simple cell responses. Thus the model suggests that both simple and some complex cells receive direct LGN input. Moreover these complex cells inhibit simple cells. With an additional threshold in simple cells this inhibition could also be obtained via complex to simple cell excitation. We stress that we are not ruling out that many complex cells receive only simple cell input. Rather, the present research shows that if all complex cells receive only simple cell input, second order motion cannot be detected. Hence at least some complex cell responses need to be built up directly from LGN responses. Several lines of evidence from cat physiology support this suggestion. First, the mean latencies of simple and complex cells are about equal (Bullier & Henry, 1979), suggesting that at least some complex cells receive direct LGN input. Second, noise stimuli can selectively activate complex cells, without activation of simple cells (Hammond, 1991). Third, cross-correlation analyses show that complex cells do receive simple cell input (Ghose et al., 1994).

The present model predicts that some cortical complex cells should respond to

second order motion. Zhou & Baker (1993) investigated this, and found that some complex cells in area 17 respond to second order motion. Moreover, they found that simple cells of a particular first order motion preference did not reverse their motion preference when stimulated with second order motion, which would occur if simple cells were just linear filters. We interpret this as further evidence that complex cells provide inhibitory input to simple cells. If complex cells are built up from LGN input, then orientation selectivity in two-dimensional space cannot be obtained based on simple cell input, but rather requires complex cells with elongated receptive fields. Thus we predict that there ought to be a correlation in complex cells between elongated receptive fields and dependence on direct LGN input.

In conclusion we have shown how the phase sensitivity of motion detectors can be mapped onto the ability to detect only first order motion, or both first and second order motion. This suggests that it is not necessary to introduce a orientation detection stage before motion detection can take place, thus simplifying the model of motion detection. Furthermore we have shown that the proposed model is in accord with known physiology.

## Acknowledgments

This work was supported by the McDonnell-Pew program in Cognitive Neuroscience.

## References

Adelson, E. & Bergen (1985). Spatiotemporal energy models for the perception of motion. *J. Opt. Soc. Am. A*, **2**, 284-299.

Bullier, J. & Henry, G. H. (1979). Ordinal position of neurons in cat striate cortex. *J. Neurophys.*, **42**, 1251-1263.

Chubb, C. & Sperling, G. (1988). Drift-balanced random stimuli: a general basis for studying non-Fourier motion perception. *J. Opt. Soc. Am. A*, **5**, 1986-2007.

Ghose, G. M., Freeman, R. D. & Ohzawa, I. (1994). Local intracortical connections in the cat's visual cortex: postnatal development and plasticity. *J. Neurophys.*, **72**, 1290-1303.

Hammond, P. (1991). On the response of simple and complex cells to random dot patterns. *Vis. Res.*, **31**, 47-50.

Hammond, P. & MacKay, D. (1983). Influence of luminance gradient reversal on simple cells in feline striate cortex. *J. Physiol.*, **337**, 69-87.

Hubel, D. H. & Wiesel, T. N. (1962). Receptive fields, binocular interaction and functional architecture in the cat's visual cortex. *J. Physiol.*, **160**, 106-154.

Poggio, T. & Reichardt, W. (1973). Considerations on models of movement detection. *Kybernetik*, **12**, 223-227.

van Santen & Sperling, G. (1985). Elaborated Reichardt detectors. *J. Opt. Soc. Am. A*, **2**, 300-321.

Wilson, H. R., Ferrera, V. P. & Yo, C. (1992). A psychophysically motivated model for two-dimensional motion perception. *Vis. Neurosci.*, **9**, 79-97.

Zhou, Y.X. & Baker, C. L. (1993). A processing stream in mammalian visual cortex neurons for non-Fourier responses. *Science*, **261**, 98-101.